# CONNECTING TO THE PAST

Bruce A. MacDonald, Assistant Professor
Knowledge Sciences Laboratory, Computer Science Department
The University of Calgary, 2500 University Drive NW
Calgary, Alberta T2N 1N4

## ABSTRACT

Recently there has been renewed interest in neural-like processing systems, evidenced for example in the two volumes *Parallel Distributed Processing* edited by Rumelhart and McClelland, and discussed as parallel distributed systems, connectionist models, neural nets, value passing systems and multiple context systems. Dissatisfaction with symbolic manipulation paradigms for artificial intelligence seems partly responsible for this attention, encouraged by the promise of massively parallel systems implemented in hardware. This paper relates simple neural-like systems based on multiple context to some other well-known formalisms—namely production systems, k-length sequence prediction, finite-state machines and Turing machines—and presents earlier sequence prediction results in a new light.

## 1  INTRODUCTION

The revival of neural net research has been very strong, exemplified recently by Rumelhart and McClelland[1], new journals and a number of meetings[a]. The nets are also described as parallel distributed systems[1], connectionist models[2], value passing systems[3] and multiple context learning systems[4,5,6,7,8,9]. The symbolic manipulation paradigm for artificial intelligence does not seem to have been as successful as some hoped[1], and there seems at last to be real promise of massively parallel systems implemented in hardware. However, in the flurry of new work it is important to consolidate new ideas and place them solidly alongside established ones. This paper relates simple neural-like systems to some other well-known notions—namely production systems, k-length sequence prediction, finite-state machines and Turing machines—and presents earlier results on the abilities of such networks in a new light.

The general form of a connectionist system[10] is simplified to a three layer net with binary fixed weights in the hidden layer, thereby avoiding many of the difficulties—and challenges—of the recent work on neural nets. The hidden unit weights are regularly patterned using a template. Sophisticated, expensive learning algorithms are avoided, and a simple method is used for determining output unit weights. In this way we gain some of the advantages of multi-layered nets, while retaining some of the simplicity of two layer net training methods. Certainly nothing is lost in computational power—as I will explain—and the limitations of two layer nets are not carried over to the simplified three layer one. Biological systems may similarly avoid the need for learning algorithms such as the "simulated annealing" method commonly used in connectionist models[11]. For one thing, biological systems do not have the same clearly distinguished training phase.

Briefly, the simplified net[b] is a production system implemented as three layers of neuron-like units; an output layer, an input layer, and a hidden layer for the productions themselves. Each hidden production unit potentially connects a predetermined set of inputs to any output. A k-length sequence predictor is formed once $k$ levels of delay unit are introduced into the input layer. k-length predictors are unable to distinguish simple sequences such as $ba\ldots a$ and $aa\ldots a$ since after $k$ or more characters the system has forgotten whether an $a$ or $b$ appeared first. If the k-length predictor is augmented with "auxiliary" actions, it is able to learn this and other regular languages, since the auxiliary actions can be equivalent to states, and can be inputs to

---

[a]Among them the 1st International Conference on Neural Nets, San Diego, CA, June 21-24, 1987, and this conference.

[b]Roughly equivalent to a single context system in Andreae's multiple context system[4,5,6,7,8,9]. See also MacDonald[12].

Figure 1: The general form of a connectionist system[10].

(a) Form of a unit

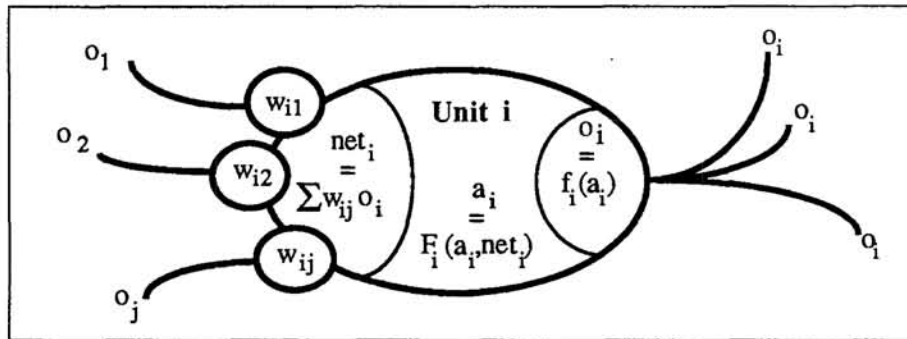

(a) Operations within a unit

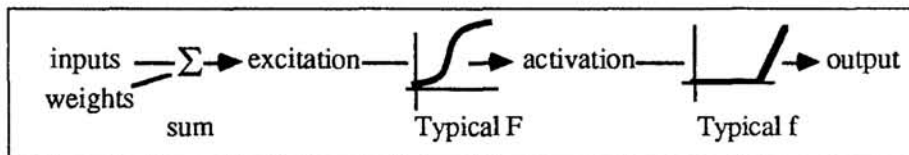

the production units enabling predictions to depend on previous states[7]. By combining several augmented sequence predictors a Turing machine tape can be simulated along with a finite-state controller[9], giving the net the computational power of a Universal Turing machine. Relatively simple neural-like systems do not lack computational ability. Previous implementations[7,9] of this ability are production system equivalents to the simplified nets.

### 1.1 Organization of the paper

The next section briefly reviews the general form of connectionist systems. Section 2 simplifies this, then section 3 explains that the result is equivalent to a production system dealing only with inputs and outputs of the net. Section 4 extends the simplified version, enabling it to learn to predict sequences. Section 5 explains how the computational power of the sequence predictor can be increased to that of a Turing machine if some input units receive auxiliary actions; in fact the system can *learn* to be a Turing machine. Section 6 discusses the possibility of a number of nets combining their outputs, forming an overall net with "association areas".

### 1.2 General form of a connectionist system

Figure 1 shows the general form of a connectionist system unit, neuron or cell[10]. In the figure unit $i$ has inputs, which are the outputs $o_j$ of possibly all units in the network, and an output of its own, $o_i$. The *net input excitation*, $net_i$, is the weighted sum of inputs, where $w_{ij}$ is the weight connecting the output from unit $j$ as an input to unit $i$. The *activation*, $a_i$ of the unit is some function $F_i$ of the net input excitation. Typically $F_i$ is *semilinear*, that is non-decreasing and differentiable[13], and is the same function for all, or at least large groups of units. The output is a function $f_i$ of the activation; typically some kind of threshold function. I will assume that the quantities vary over discrete time steps, so for example the activation at time $t + 1$ is $a_i(t + 1)$ and is given by $F_i((net_i(t))$.

In general there is no restriction on the connections that may be made between units. Units not connected directly to inputs or outputs are *hidden* units. In more complex nets than those described in this paper, there may be more than one *type* of connection. Figure 2 shows a common connection topology, where there are three layers of units—input, hidden and output—with no cycles of connection.

The net is trained by presenting it with input combinations, each along with the desired output combination. Once trained the system should produce the desired outputs given just

Figure 2: The basic structure of a three layer connectionist system.

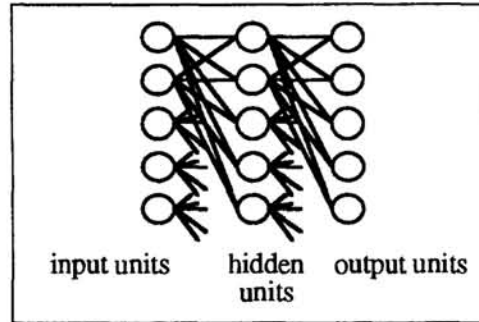

inputs. During training the weights are adjusted in some fashion that reduces the discrepancy between desired and actual output. The general method is[10]:

$$\Delta w_{ij} = g(a_i, t_i)\, h(o_j, w_{ij}), \qquad (1)$$

where $t_i$ is the desired, "training" activation. Equation 1 is a general form of Hebb's classic rule for adjusting the weight between two units with high activations[10]. The weight adjustment is the product of two functions, one that depends on the desired and actual activations—often just the difference—and another that depends on the input to that weight and the weight itself. As a simple example suppose $g$ is the difference and $h$ as just the output $o_j$. Then the weight change is the product of the output error and the input excitation to that weight:

$$\Delta w_{ij} = \eta o_j (t_i - a_i)$$

where the constant $\eta$ determines the learning rate. This is the Widrow-Hoff or Delta rule which may be used in nets without hidden units.[10]

The important contribution of recent work on connectionist systems is how to implement equation 1 in hidden units; for which there are no training signals $t_i$ directly available. The Boltzmann learning method iteratively varies both weights and hidden unit training activations using the controlled, gradually decreasing randomizing method "simulated annealing"[14]. Back-propagation[13] is also iterative, performing gradient descent by propagating training signal errors back through the net to hidden units. I will avoid the need to determine training signals for hidden units, by fixing the weights of hidden units in section 2 below.

## 2 SIMPLIFIED SYSTEM

Assume these simplifications are made to the general connectionist system of section 1.2:

1. The system has three layers, with the topology shown in Figure 2 (ie no cycles)

2. *All* hidden layer unit weights are fixed, say at unity or zero

3. Each unit is a linear threshold unit[10], which means the activation function for all units is the identity function, giving just $net_i$, a weighted sum of the inputs, and the output function is a simple binary threshold of the form:

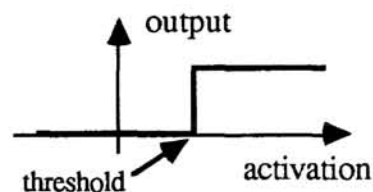

so that the output is binary; on or off. Hidden units will have thresholds requiring all inputs to be active for the output to be active (like an AND gate) while output units will have thresholds requiring only 1 or two active highly weighted inputs for an output to be generated (like an OR gate). This is in keeping with the production system view of the net, explained in section 3.

4. Learning—which now occurs only at the output unit weights—gives weight adjustments according to:

$$w_{ij} = 1 \quad if \ a_i = o_j = 1$$
$$w_{ij} = 0 \quad \text{otherwise}$$

so that weights are turned on if their input and the unit output are on, and off otherwise. That is, $w_{ij} = a_i \wedge o_j$. A simple example is given in Figure 3 in section 3 below.

This simple form of net can be made probabilistic by replacing 4 with 4' below:

4'. Adjust weights so that $w_{ij}$ estimates the conditional probability of the unit $i$ output being on when output $j$ is on. That is,

$$w_{ij} = \text{estimate of } P(o_i|o_j).$$

Then, assuming independence of the inputs to a unit, an output unit is turned on when the conditional probability of occurrence of that output exceeds the threshold of the output function.

Once these simplifications are made, there is no need for learning in the hidden units. Also no iterative learning is required; weights are either assigned binary values, or estimate conditional probabilities. This paper presents some of the characteristics of the simplified net. Section 6 discusses the motivation for simplifying neural nets in this way.

## 3   PRODUCTION SYSTEMS

The simplified net is a kind of simple production system. A production system comprises a global database, a set of production rules and a control system[15]. The database for the net is the system it interacts with, providing inputs as reactions to outputs from the net. The hidden units of the network are the production rules, which have the form

IF *precondition* THEN *action*

The precondition is satisfied when the input excitation exceeds the threshold of a hidden unit. The actions are represented by the output units which the hidden production units activate. The control system of a production system chooses the rule whose action to perform, from the set of rules whose preconditions have been met. In a neural net the control system is distributed throughout the net in the output units. For example, the output units might form a winner-take-all net. In production systems more complex control involves forward and backward chaining to choose actions that seek goals. This is discussed elsewhere[4,12,16]. Figure 3 illustrates a simple production implemented as a neural net. As the figure shows, the inputs to hidden units are just the elements of the precondition. When the appropriate input combination is present the associated hidden (production) unit is fired. Once weights have been learned connecting hidden units to output units, firing a production results in output. The simplified neural net is directly equivalent to a production system whose elements are inputs and outputs[c].

Some production systems have symbolic elements, such as variables, which can be given values by production actions. The neural net cannot directly implement this, since it can have outputs only from a predetermined set. However, we will see later that extensions to the framework enable this and other abilities.

---

[c]This might be referred to as a "sensory-motor" production system, since when implemented in a real system such as a robot, it deals only with sensed inputs and executable motor actions, which may include the auxiliary actions of section 4.3.

Figure 3: A production implemented in a simplified neural net.

(a) A production rule

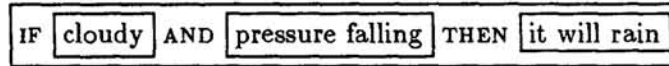

(b) The rule implemented as a hidden unit. The threshold of the hidden unit is 2 so it is
an AND gate. The threshold of the output unit is 1 so it is an OR gate. The learned
weight will be 0 or 1 if the net is not probabilistic, otherwise it will be an estimate of
$P$(it will rain|clouds AND pressure falling)

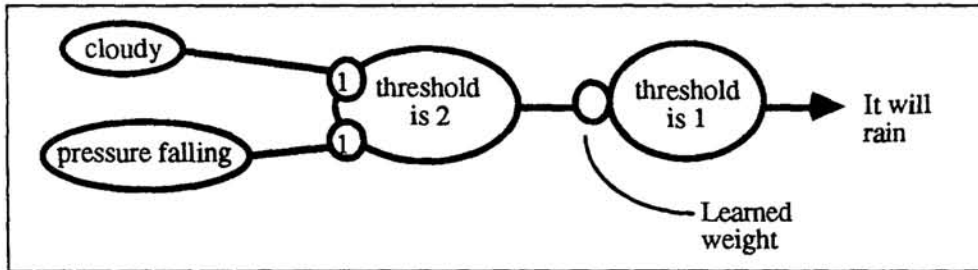

Figure 4: A net that predicts the next character in a sequence, based on only the last character.

(a) The net. Production units (hidden units) have been combined with input units.
For example this net could predict the sequence *abcabcabc*.... Productions have the
form: IF last character is ...THEN next character will be .... The learning rule is
$w_{ij} = 1$ if (input$_j$ AND output$_i$). Output is $a_i = \underset{j}{OR} \ w_{ij}o_j$

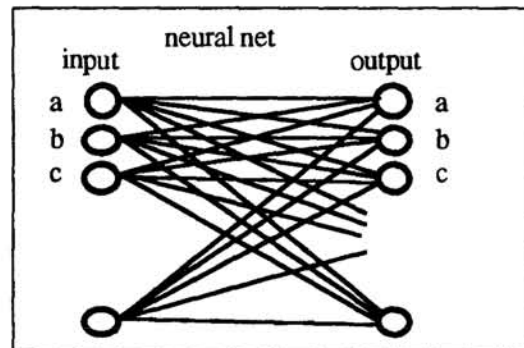

(b) Learning procedure.

1. Clamp inputs and outputs to desired values

2. System calculates weight values

3. Repeat 4 and 4 for all required input/output combinations

## 4  SEQUENCE PREDICTION

A production system or neural net can predict sequences. Given examples of a repeating se-
quence, productions are learned which predict future events on the basis of recent ones. Figure 4
shows a trivially simple sequence predictor. It predicts the next character of a sequence based
on the previous one. The figure also gives the details of the learning procedure for the simplified
net. The net need be trained only once on each input combination, then it will "predict" as
an output every character seen after the current one. The probabilistic form of the net would
estimate conditional probabilities for the next character, conditional on the current one. Many

Figure 5: Using delayed inputs, a neural net can implement a k-length sequence predictor. (a) A net with the last three characters as input.

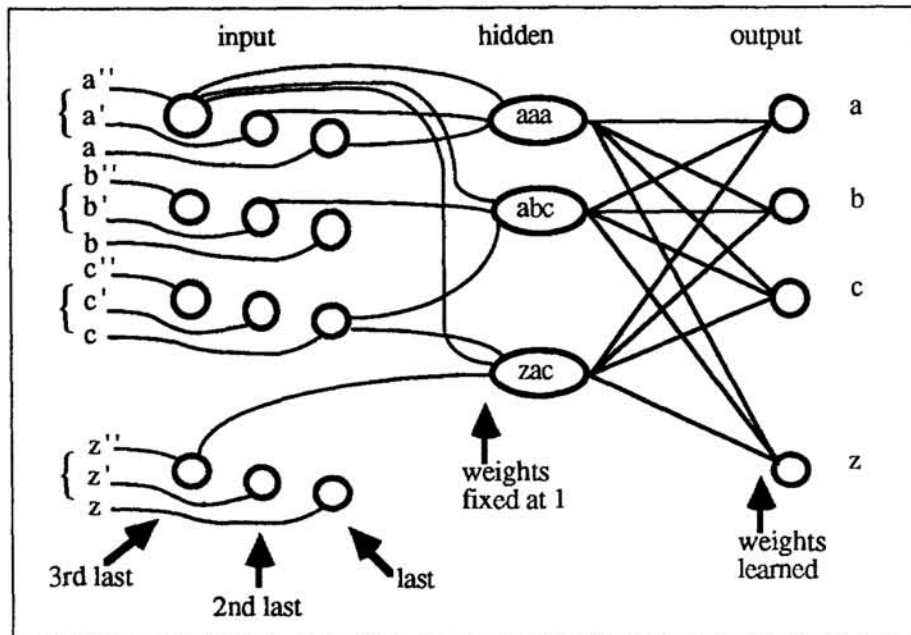

(b) An example production.

IF last three characters were $\boxed{\text{abc}}$ THEN $\boxed{\text{a}}$

presentations of each possible character pair would be needed to properly estimate the probabilities. The net would be learning the probability distribution of character pairs. A predictor like the one in Figure 4 can be extended to a general k-length[17] predictor so long as inputs delayed by $1, 2, \ldots, k$ steps are available. Then, as illustrated in Figure 5 for 3-length prediction, hidden production units represent all possible combinations of $k$ symbols. Again output weights are trained to respond to previously seen input combinations, here of three characters. These delays can be provided by dedicated neural nets[d], such as that shown in Figure 6. Note that the net is assumed to be synchronously updated, so that the input from feedback around units is not changed until one step after the output changes. There are various ways of implementing delay in neurons, and Andreae[4] investigates some of them for the same purpose—delaying inputs—in a more detailed simulation of a similar net.

### 4.1 Other work on sequence prediction in neural nets

Feldman and Ballard[2] find connectionist systems initially not suited to representing changes with time. One form of change is sequence, and they suggest two methods for representing sequence in nets. The first is by units connected to each other in sequence so that sequential tasks are represented by firing these units in succession. The second method is to buffer the inputs in time so that inputs from the recent past are available as well as current inputs; that is, delayed inputs are available as suggested above. An important difference is the necessary length of the buffer; Feldman and Ballard suggest the buffer be long enough to hold a phrase of natural language, but I expect to use buffers no longer than about 7, after Andreae[4]. Symbolic inputs can represent more complex information effectively giving the length seven buffers more information than the most recent seven simple inputs, as discussed in section 5.

The method of back-propagation[13] enables recurrent networks to learn sequential tasks in a

---

[d]Feldman and Ballard[2] give some dedicated neural net connections for a variety of functions

Figure 6: Inputs can be delayed by dedicated neural subnets. A two stage delay is shown.
(a) Delay network.

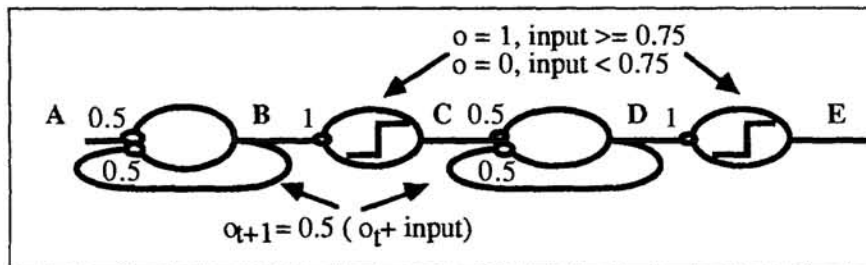

(b) Timing diagram for (a).

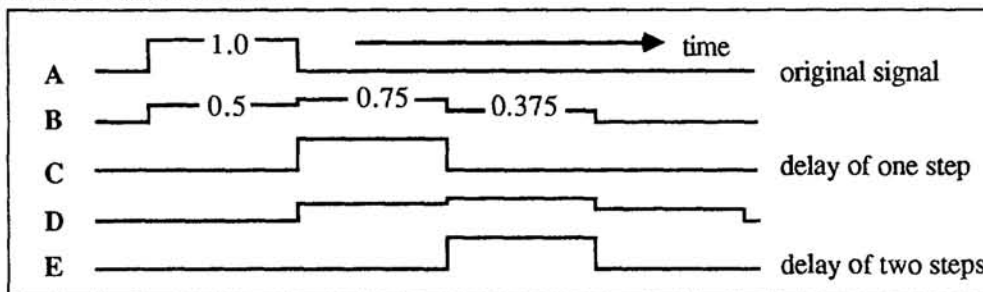

manner similar to the first suggestion in the last paragraph, where sequences of connected units represent sequenced events. In one example a net learns to complete a sequence of characters; when given the first two characters of a six character sequence the next four are output. Errors must be propagated around cycles in a recurrent net a number of times.

Seriality may also be achieved by a sequence of states of distributed activation[18]. An example is a net playing both sides of a tic-tac-toe game[18]. The sequential nature of the net's behavior is derived from the sequential nature of the responses to the net's actions; tic-tac-toe moves. A net can model sequence internally by modeling a sequential part of its environment. For example, a tic-tac-toe playing net can have a model of its opponent.

k-length sequence predictors are unable to learn sequences which do not repeat more frequently that every $k$ characters. Their k-length context includes only information about the last $k$ events. However, there are two ways in which information from before the $kth$ last input can be retained in the net. The first method latches some inputs, while the second involves auxiliary actions.

### 4.2   Latch units

Inputs can be latched and held indefinitely using the combination shown in Figure 7. Not all inputs would normally be latched. Andreae[4] discusses this technique of "threading" latched events among non-latched events, giving the net both information arbitrarily far back in its input-output history and information from the immediate past. Briefly, the sequence $ba \ldots a$ can be distinguished from $aa \ldots a$ if the first character is latched. However, this is an *ad hoc* solution to this problem[e].

### 4.3   Auxiliary actions

When an output is fed back into the net as an input signal, this enables the system to choose the next output at least partly based on the previous one, as indicated in Figure 8. If a particular fed back output is also one without external manifestation, or whose external manifestation is independent of the task being performed, then that output is an *auxiliary* action. It has

---

[e]The interested reader should refer to Andreae[4] where more extensive analysis is given.

Figure 7: Threading. A latch circuit remembers an event until another comes along. This is a two input latch, e.g. for two letters $a$ and $b$, but any number of units may be similarly connected. It is formed from a mutual inhibition layer, or winner-take-all connection, along with positive feedback to keep the selected output activated when the input disappears.

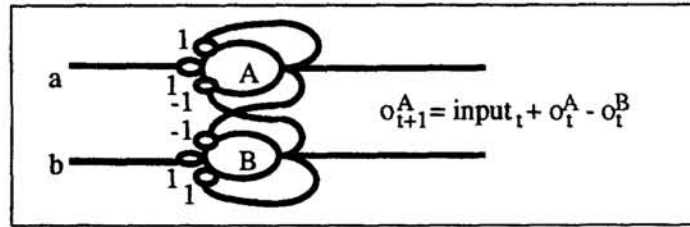

$$o_{t+1}^A = \text{input}_t + o_t^A - o_t^B$$

Figure 8: Auxiliary actions—the S outputs—are fed back to the inputs of a net, enabling the net to remember a state. Here both part of a net and an example of a production are shown. There are two types of action, characters and S actions.

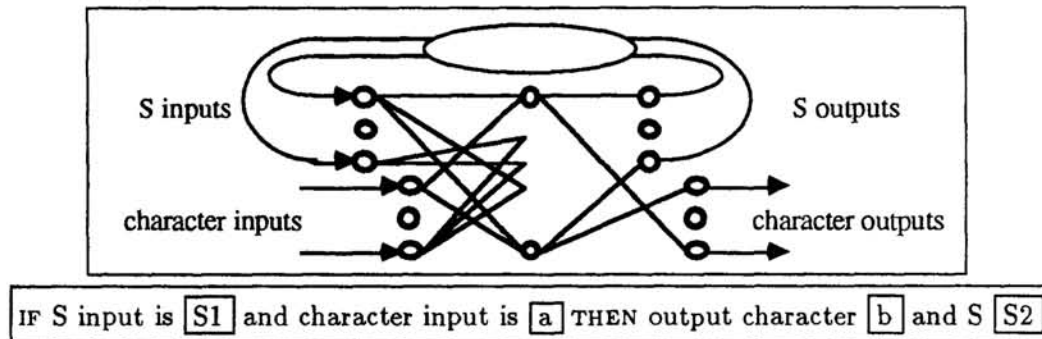

IF S input is [S1] and character input is [a] THEN output character [b] and S [S2]

no direct effect on the task the system is performing since it evokes no relevant inputs, and so can be used by the net as a *symbolic* action. If an auxiliary action is latched at the input then the symbolic information can be remembered indefinitely, being lost only when another auxiliary action of that kind is input and takes over the latch. Thus auxiliary actions can act like remembered states; the system performs an action to "remind" itself to be in a particular state. The figure illustrates this for a system that predicts characters and state changes given the previous character and state. An obvious candidate for auxiliary actions is speech. So the blank oval in the figure would represent the net's environment, through which its own speech actions are heard. Although it is externally manifested, speech has no direct effect on our physical interactions with the world. Its symbolic ability not only provides the power of auxiliary actions, but also includes other speakers in the interaction.

## 5  SIMULATING ABSTRACT AUTOMATA

The example in Figure 8 gives the essence of simulating a finite state automaton with a production system or its neural net equivalent. It illustrates the transition function of an automaton; the new state and output are a function of the previous state and input. Thus a neural net can simulate a finite state automaton, so long as it has additional, auxiliary actions.

A Turing machine is a finite state automaton controller plus an unbounded memory. A neural net could simulate a Turing machine in two ways, and both ways have been demonstrated with production system implementations—equivalent to neural nets—called "multiple context learning systems"[f], briefly explained in section 6. The first Turing machine simulation[7] has the system simulate only the finite state controller, but is able to use an unbounded external memory

---

[f]See John Andreae's and his colleagues' work[4,5,6,7,8,9,12,16]

Figure 9: Multiple context learning system implementation as multiple neural nets. Each 3 layer net has the simplified form presented above, with a number of elaborations such as extra connections for goal-seeking by forward and backward chaining.

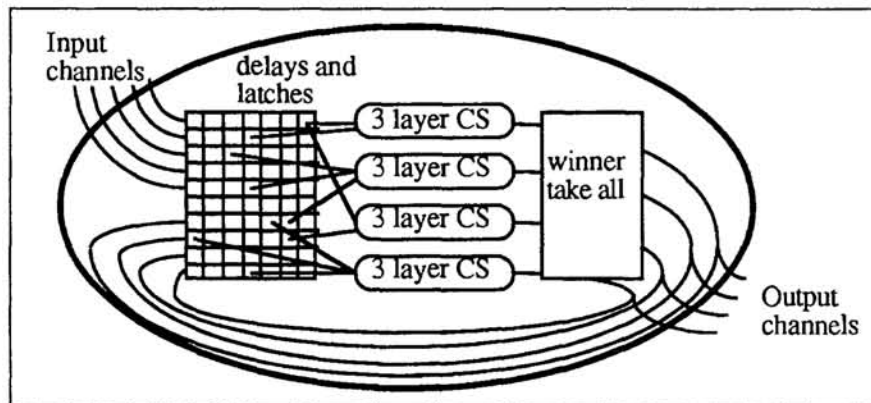

from the real world, much like the paper of Turing's original work[19]. The second simulation[9,12] embeds the memory in the multiple context learning system, along with a counter for accessing this simulated memory. Both learn all the productions—equivalent to learning output unit weights—required for the simulations. The second is able to add internal memory as required, up to a limit dependent on the size of the network (which can easily be large enough to allow 70 years of computation!). The second could also employ external memory as the first did. Briefly, the second simulation comprised multiple sequence predictors which predicted auxiliary actions for remembering the state of the controller, and the current memory position. The memory element is updated by relearning the production representing that element; the precondition is the address and the production action the stored item.

## 6   MULTIPLE SYSTEMS FORM ASSOCIATION AREAS

A multiple context learning system is production system version of a multiple neural net, although a simple version has been implemented as a simulated net[4,20]. It effectively comprises several nets—or "association" areas—which may have outputs and inputs in common, as indicated in Figure 9. Hidden unit weights are specified by templates; one for each net. A template gives the inputs to have a zero weight for the hidden units of a net and the inputs to have a weight of unity. Delayed and latched inputs are also available. The actual outputs are selected from the combined predictions of the nets in a winner-take-all fashion.

I see the design for real neural nets, say as controllers for real robots, requiring a large degree of predetermined connectivity. A robot controller could not be one three layer net with every input connected to every hidden unit in turn connected to every output. There will need to be some connectivity constraints so the net reflects the functional specialization in the control requirements[9]. The multiple context learning system has all the hidden layer connections predetermined, but allows output connections to be learned. This avoids the "credit assignment" problem and therefore also the need for learning algorithms such as Boltzmann learning and back-propagation. However, as the multiple context learning system has auxiliary actions, and delayed and latched inputs, it does not lack computational power. Future work in this area should investigate, for example, the ability of different kinds of nets to learn auxiliary actions. This may be difficult as symbolic actions may not be provided in training inputs and outputs.

## 7 CONCLUSION

This paper has presented a simplified three layer connectionist model, with fixed weights for hidden units, delays and latches for inputs, sequence prediction ability, auxiliary "state" actions, and the ability to use internal and external memory. The result is able to learn to simulate a Turing machine. Simple neural-like systems do not lack computational power.

## ACKNOWLEDGEMENTS

This work is supported by the Natural Sciences and Engineering Council of Canada.

## Footnotes

[9]For example a controller for a robot body would have to deal with vision, manipulation, motion, etc.

## REFERENCES

1. Rumelhart,D.E. and McClelland,J.L. Parallel distributed processing. Volumes 1 and 2. MIT Press. (1986)
2. Feldman,J.A. and Ballard,D.H. Connectionist models and their properties. *Cognitive Science 6*, pp.205-254. (1982)
3. Fahlman,S.E. Three Flavors of Parallelism. Proc.4th Nat.Conf. CSCSI/SCSEIO, Saskatoon. (1982)
4. Andreae,J.H. Thinking with the teachable machine. Academic Press. (1977)
5. Andreae,J.H. Man-Machine Studies Progress Reports UC-DSE/1-28. Dept Electrical and Electronic Engineering, Univ. Canterbury, Christchurch, New Zealand. editor. (1972-87) (Also available from NTIS, 5285 Port Royal Rd, Springfield, VA 22161)
6. Andreae,J.H. and Andreae,P.M. Machine learning with a multiple context. Proc.9th Int.Conf.on Cybernetics and Society. Denver. October. pp.734-9. (1979)
7. Andreae,J.H. and Cleary,J.G. A new mechanism for a brain. *Int.J.Man-Machine Studies 8*(1): pp.89-119. (1976)
8. Andreae,P.M. and Andreae,J.H. A teachable machine in the real world. *Int.J.Man-Machine Studies 10*: pp.301-12. (1978)
9. MacDonald,B.A. and Andreae,J.H. The competence of a multiple context learning system. *Int.J.Gen.Systems 7*: pp.123-37. (1981)
10. Rumelhart,D.E., Hinton,G.E. and McClelland,J.L. A general framework for parallel distributed processing. chapter 2 in Rumelhart and McClelland[1], pp.45-76. (1986)
11. Hinton,G.E. and Sejnowski,T.L. Learning and relearning in Boltzmann machines. chapter 7 in Rumelhart and McClelland[1], pp.282-317. (1986)
12. MacDonald,B.A. Designing teachable robots. PhD thesis, University of Canterbury, Christchurch, New Zealand. (1984)
13. Rumelhart,D.E., Hinton,G.E. and Williams,R.J. Learning Internal Representations by Error Propagation. chapter 8 in Rumelhart and McClelland[1], pp.318-362. (1986)
14. Ackley,D.H., Hinton,G.E. and Sejnowski,T.J. A Learning Algorithm for Boltzmann Machines. *Cognitive Science 9*, pp.147-169. (1985)
15. Nilsson,N.J. Principles of Artificial Intelligence. Tioga. (1980)
16. Andreae,J.H. and MacDonald,B.A. Expert control for a robot body. Research Report 87/286/34 Dept. of Computer Science, University of Calgary, Alberta, Canada, T2N-1N4. (1987)
17. Witten,I.H. Approximate, non-deterministic modelling of behaviour sequences. *Int. J. General Systems, vol. 5* pp.1-12. (1979)
18. Rumelhart,D.E.,Smolensky,P.,McClelland,J.L. and Hinton,G.E. Schemata and Sequential thought Processes in PDP Models. chapter 14, vol 2 in Rumelhart and McClelland[1]. pp.7-57. (1986)
19. Turing,A.M. On computable numbers, with an application to the entscheidungsproblem. *Proc. London Math. Soc. vol 42*(3). pp. 230-65. (1936)
20. Dowd,R.B. A digital simulation of mew-brain. Report no. UC-DSE/10⁵. pp.25-46. (1977)
